# Multimodal Learning with Deep Boltzmann Machines

**Nitish Srivastava**
Department of Computer Science
University of Toronto
nitish@cs.toronto.edu

**Ruslan Salakhutdinov**
Department of Statistics and Computer Science
University of Toronto
rsalakhu@cs.toronto.edu

## Abstract

A Deep Boltzmann Machine is described for learning a generative model of data that consists of multiple and diverse input modalities. The model can be used to extract a unified representation that fuses modalities together. We find that this representation is useful for classification and information retrieval tasks. The model works by learning a probability density over the space of multimodal inputs. It uses states of latent variables as representations of the input. The model can extract this representation even when some modalities are absent by sampling from the conditional distribution over them and filling them in. Our experimental results on bi-modal data consisting of images and text show that the Multimodal DBM can learn a good generative model of the joint space of image and text inputs that is useful for information retrieval from both unimodal and multimodal queries. We further demonstrate that this model significantly outperforms SVMs and LDA on discriminative tasks. Finally, we compare our model to other deep learning methods, including autoencoders and deep belief networks, and show that it achieves noticeable gains.

## 1 Introduction

Information in the real world comes through multiple input channels. Images are associated with captions and tags, videos contain visual and audio signals, sensory perception includes simultaneous inputs from visual, auditory, motor and haptic pathways. Each modality is characterized by very distinct statistical properties which make it difficult to ignore the fact that they come from different input channels. Useful representations can be learned about such data by fusing the modalities into a joint representation that captures the real-world 'concept' that the data corresponds to. For example, we would like a probabilistic model to correlate the occurrence of the words 'beautiful sunset' and the visual properties of an image of a beautiful sunset and represent them jointly, so that the model assigns high probability to one conditioned on the other.

Before we describe our model in detail, it is useful to note why such a model is required. Different modalities typically carry different kinds of information. For example, people often caption an image to say things that may not be obvious from the image itself, such as the name of the person, place, or a particular object in the picture. Unless we do multimodal learning, it would not be possible to discover a lot of useful information about the world (for example, 'what do beautiful sunsets look like?'). We cannot afford to have discriminative models for each such concept and must extract this information from unlabeled data.

In a multimodal setting, data consists of multiple input modalities, each modality having a different kind of representation and correlational structure. For example, text is usually represented as discrete sparse word count vectors, whereas an image is represented using pixel intensities or outputs of feature extractors which are real-valued and dense. This makes it much harder to discover relationships across modalities than relationships among features in the same modality. There is a lot of structure in the input but it is difficult to discover the highly non-linear relationships that exist

| Image | Given Tags | Generated Tags | Input Text | 2 nearest neighbours to generated image features |
|---|---|---|---|---|
|  | pentax, k10d, kangarooisland, southaustralia, sa, australia, australiansealion, 300mm | beach, sea, surf, strand, shore, wave, seascape, sand, ocean, waves | nature, hill scenery, green clouds |   |
|  | \<no text\> | night, lights, christmas, nightshot, nacht, nuit,notte, longexposure, noche, nocturna | flower, nature, green, flowers, petal, petals, bud |   |
|  | aheram, 0505 sarahc, moo | portrait, bw, blackandwhite, woman, people, faces, girl,blackwhite, person, man | blue, red, art, artwork, painted, paint, artistic surreal, gallery bleu |   |
| 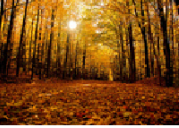 | unseulpixel, naturey crap | fall, autumn, trees, leaves, foliage, forest, woods, branches, path | bw, blackandwhite, noiretblanc, biancoenero blancoynegro | 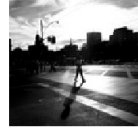  |

Figure 1: **Left:** Examples of text generated from a DBM by sampling from $P(\mathbf{v}_{txt}|\mathbf{v}_{img}, \theta)$. **Right:** Examples of images retrieved using features generated from a DBM by sampling from $P(\mathbf{v}_{img}|\mathbf{v}_{txt}, \theta)$.

between low-level features across different modalities. Moreover, these observations are typically very noisy and may have missing values.

A good multimodal learning model must satisfy certain properties. The joint representation must be such that similarity in the representation space implies similarity of the corresponding 'concepts'. It is also desirable that the joint representation be easy to obtain even in the absence of some modalities. It should also be possible to fill-in missing modalities given the observed ones. In addition, it is also desirable that extracted representation be useful for discriminative tasks.

Our proposed multimodal Deep Boltzmann Machine (DBM) model satisfies the above desiderata. DBMs are undirected graphical models with bipartite connections between adjacent layers of hidden units [1]. The key idea is to learn a joint density model over the space of multimodal inputs. Missing modalities can then be filled-in by sampling from the conditional distributions over them given the observed ones. For example, we use a large collection of user-tagged images to learn a joint distribution over images and text $P(\mathbf{v}_{img}, \mathbf{v}_{txt}|\theta)$. By drawing samples from $P(\mathbf{v}_{txt}|\mathbf{v}_{img}, \theta)$ and from $P(\mathbf{v}_{img}|\mathbf{v}_{txt}, \theta)$ we can fill-in missing data, thereby doing image annotation and image retrieval respectively, as shown in Fig. 1.

There have been several approaches to learning from multimodal data. In particular, Huiskes et al. [2] showed that using captions, or tags, in addition to standard low-level image features significantly improves classification accuracy of Support Vector Machines (SVM) and Linear Discriminant Analysis (LDA) models. A similar approach of Guillaumin et al. [3], based on multiple kernel learning framework, further demonstrated that an additional text modality can improve the accuracy of SVMs on various object recognition tasks. However, all of these approaches are discriminative by nature and cannot make use of large amounts of unlabeled data or deal easily with missing input modalities.

On the generative side, Xing et al. [4] used dual-wing harmoniums to build a joint model of images and text, which can be viewed as a linear RBM model with Gaussian hidden units together with Gaussian and Poisson visible units. However, various data modalities will typically have very different statistical properties which makes it difficult to model them using shallow models. Most similar to our work is the recent approach of Ngiam et al. [5] that used a deep autoencoder for speech and vision fusion. There are, however, several crucial differences. First, in this work we focus on integrating together very different data modalities: sparse word count vectors and real-valued dense image features. Second, we develop a Deep Boltzmann Machine as a generative model as opposed to unrolling the network and fine-tuning it as an autoencoder. While both approaches have lead to interesting results in several domains, using a generative model is important for applications we consider in this paper, as it allows our model to naturally handle missing data modalities.

## 2   Background: RBMs and Their Generalizations

Restricted Boltzmann Machines (RBMs) have been used effectively in modeling distributions over binary-valued data. Recent work on Boltzmann machine models and their generalizations to exponential family distributions have allowed these models to be successfully used in many application domains. In particular, the Replicated Softmax model [6] has been shown to be effective in modeling sparse word count vectors, whereas Gaussian RBMs have been used for modeling real-valued inputs for speech and vision tasks. In this section we briefly review these models, as they will serve as our building blocks for the multimodal model.

### 2.1   Restricted Boltzmann Machines

A Restricted Boltzmann Machine is an undirected graphical model with stochastic visible units $\mathbf{v} \in \{0,1\}^D$ and stochastic hidden units $\mathbf{h} \in \{0,1\}^F$, with each visible unit connected to each hidden unit. The model defines the following energy function $E : \{0,1\}^{D+F} \to \mathbb{R}$:

$$E(\mathbf{v}, \mathbf{h}; \theta) = -\sum_{i=1}^{D} \sum_{j=1}^{F} v_i W_{ij} h_j - \sum_{i=1}^{D} b_i v_i - \sum_{j=1}^{F} a_j h_j$$

where $\theta = \{\mathbf{a}, \mathbf{b}, \mathbf{W}\}$ are the model parameters. The joint distribution over the visible and hidden units is defined by:

$$P(\mathbf{v}, \mathbf{h}; \theta) = \frac{1}{\mathcal{Z}(\theta)} \exp\left(-E(\mathbf{v}, \mathbf{h}; \theta)\right). \tag{1}$$

### 2.2   Gaussian RBM

Consider modeling visible real-valued units $\mathbf{v} \in \mathbb{R}^D$, and let $\mathbf{h} \in \{0,1\}^F$ be binary stochastic hidden units. The energy of the state $\{\mathbf{v}, \mathbf{h}\}$ of the Gaussian RBM is defined as follows:

$$E(\mathbf{v}, \mathbf{h}; \theta) = \sum_{i=1}^{D} \frac{(v_i - b_i)^2}{2\sigma_i^2} - \sum_{i=1}^{D} \sum_{j=1}^{F} \frac{v_i}{\sigma_i} W_{ij} h_j - \sum_{j=1}^{F} a_j h_j, \tag{2}$$

where $\theta = \{\mathbf{a}, \mathbf{b}, \mathbf{W}, \sigma\}$ are the model parameters.

### 2.3   Replicated Softmax Model

The Replicated Softmax Model is useful for modeling sparse count data, such as word count vectors in a document. Let $\mathbf{v} \in \mathbb{N}^K$ be a vector of visible units where $v_k$ is the number of times word $k$ occurs in the document with the vocabulary of size $K$. Let $\mathbf{h} \in \{0,1\}^F$ be binary stochastic hidden topic features. The energy of the state $\{\mathbf{v}, \mathbf{h}\}$ is defined as follows

$$E(\mathbf{v}, \mathbf{h}; \theta) = -\sum_{k=1}^{K} \sum_{j=1}^{F} v_k W_{kj} h_j - \sum_{k=1}^{K} b_k v_k - M \sum_{j=1}^{F} a_j h_j \tag{3}$$

where $\theta = \{\mathbf{a}, \mathbf{b}, \mathbf{W}\}$ are the model parameters and $M = \sum_k v_k$ is the total number of words in a document. We note that this replicated softmax model can also be interpreted as an RBM model that uses a single visible multinomial unit with support $\{1, ..., K\}$ which is sampled $M$ times.

For all of the above models, exact maximum likelihood learning is intractable. In practice, efficient learning is performed using Contrastive Divergence (CD) [7].

## 3   Multimodal Deep Boltzmann Machine

A Deep Boltzmann Machine (DBM) is a network of symmetrically coupled stochastic binary units. It contains a set of visible units $\mathbf{v} \in \{0,1\}^D$, and a sequence of layers of hidden units $\mathbf{h}^{(1)} \in \{0,1\}^{F_1}$, $\mathbf{h}^{(2)} \in \{0,1\}^{F_2}$,..., $\mathbf{h}^{(L)} \in \{0,1\}^{F_L}$. There are connections only between hidden units in adjacent layers. Let us first consider a DBM with two hidden layers. The energy of the joint configuration $\{\mathbf{v}, \mathbf{h}\}$ is defined as (ignoring bias terms):

$$E(\mathbf{v}, \mathbf{h}; \theta) = -\mathbf{v}^\top \mathbf{W}^{(1)} \mathbf{h}^{(1)} - \mathbf{h}^{(1)\top} \mathbf{W}^{(2)} \mathbf{h}^{(2)},$$

where $\mathbf{h} = \{\mathbf{h}^{(1)}, \mathbf{h}^{(2)}\}$ represent the set of hidden units, and $\theta = \{\mathbf{W}^{(1)}, \mathbf{W}^{(2)}\}$ are the model parameters, representing visible-to-hidden and hidden-to-hidden symmetric interaction terms. Similar to RBMs, this binary-binary DBM can be easily extended to modeling dense real-valued or sparse count data, which we discuss next.

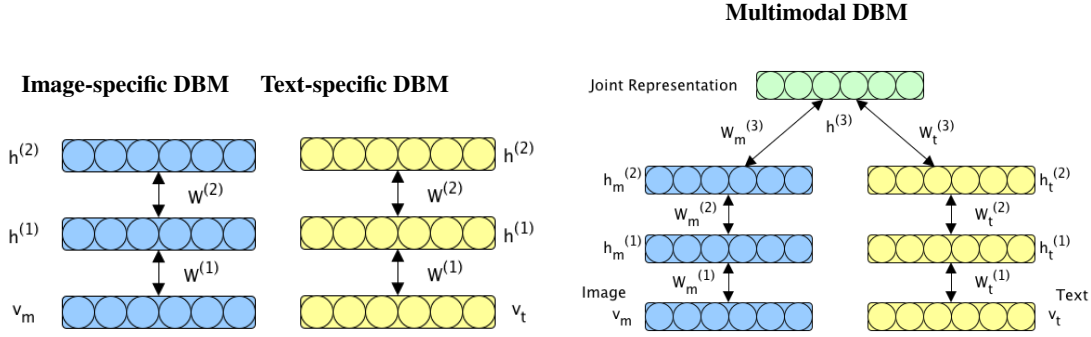

Figure 2: **Left:** Image-specific two-layer DBM that uses a Gaussian model to model the distribution over real-valued image features. **Middle:** Text-specific two-layer DBM that uses a Replicated Softmax model to model its distribution over the word count vectors. **Right:** A Multimodal DBM that models the joint distribution over image and text inputs.

We illustrate the construction of a multimodal DBM using an image-text bi-modal DBM as our running example. Let $\mathbf{v}_m \in \mathbb{R}^D$ denote an image input and $\mathbf{v}_t \in \mathbb{N}^K$ denote a text input. Consider modeling each data modality using separate two-layer DBMs (Fig. 2). The image-specific two-layer DBM assigns probability to $\mathbf{v}_m$ that is given by (ignoring bias terms on the hidden units for clarity):

$$P(\mathbf{v}_m; \theta) = \sum_{\mathbf{h}^{(1)}, \mathbf{h}^{(2)}} P(\mathbf{v}_m, \mathbf{h}^{(1)}, \mathbf{h}^{(2)}; \theta) = \tag{4}$$

$$= \frac{1}{\mathcal{Z}(\theta)} \sum_{\mathbf{h}^{(1)}, \mathbf{h}^{(2)}} \exp\left( -\sum_{i=1}^{D} \frac{(v_{mi} - b_i)^2}{2\sigma_i^2} + \sum_{i=1}^{D}\sum_{j=1}^{F_1} \frac{v_{mi}}{\sigma_i} W_{ij}^{(1)} h_j^{(1)} + \sum_{j=1}^{F_1}\sum_{l=1}^{F_2} h_j^{(1)} W_{jl}^{(2)} h_l^{(2)} \right).$$

Note that we borrow the visible-hidden interaction term from the Gaussian RBM (Eq. 2) and the hidden-hidden one from the Binary RBM (Eq. 1). Similarly, the text-specific DBM will use terms from the Replicated Softmax model for the visible-hidden interactions (Eq. 3) and the hidden-hidden ones from the Binary RBM (Eq. 1).

To form a multimodal DBM, we combine the two models by adding an additional layer of binary hidden units on top of them. The resulting graphical model is shown in Fig. 2, right panel. The joint distribution over the multi-modal input can be written as:

$$P(\mathbf{v}^m, \mathbf{v}^t; \theta) = \sum_{\mathbf{h}_m^{(2)}, \mathbf{h}_t^{(2)}, \mathbf{h}^{(3)}} P(\mathbf{h}_m^{(2)}, \mathbf{h}_t^{(2)}, \mathbf{h}^{(3)}) \left( \sum_{\mathbf{h}_m^{(1)}} P(\mathbf{v}_m, \mathbf{h}_m^{(1)}, \mathbf{h}_m^{(2)}) \right) \left( \sum_{\mathbf{h}_t^{(1)}} P(\mathbf{v}_t, \mathbf{h}_t^{(1)}, \mathbf{h}_t^{(2)}) \right).$$

### 3.1 Approximate Learning and Inference

Exact maximum likelihood learning in this model is intractable, but efficient approximate learning can be carried out by using mean-field inference to estimate data-dependent expectations, and an MCMC based stochastic approximation procedure to approximate the model's expected sufficient statistics [1]. In particular, during the inference step, we approximate the true posterior $P(\mathbf{h}|\mathbf{v}; \theta)$, where $\mathbf{v} = \{\mathbf{v}_m, \mathbf{v}_t\}$, with a fully factorized approximating distribution over the five sets of hidden units $\{\mathbf{h}_m^{(1)}, \mathbf{h}_m^{(2)}, \mathbf{h}_t^{(1)}, \mathbf{h}_t^{(2)}, \mathbf{h}^{(3)}\}$:

$$Q(\mathbf{h}|\mathbf{v}; \boldsymbol{\mu}) = \left( \prod_{j=1}^{F_1} q(h_{mj}^{(1)}|\mathbf{v}) \prod_{l=1}^{F_2} q(h_{ml}^{(2)}|\mathbf{v}) \right) \left( \prod_{j=1}^{F_1} q(h_{tj}^{(1)}|\mathbf{v}) \prod_{l=1}^{F_2} q(h_{tl}^{(2)}|\mathbf{v}) \right) \prod_{k=1}^{F_3} q(h_k^{(3)}|\mathbf{v}), \tag{5}$$

where $\boldsymbol{\mu} = \{\boldsymbol{\mu}_m^{(1)}, \boldsymbol{\mu}_m^{(2)}, \boldsymbol{\mu}_t^{(1)}, \boldsymbol{\mu}_t^{(2)}, \boldsymbol{\mu}^{(3)}\}$ are the mean-field parameters with $q(h_i^{(l)} = 1) = \mu_i^{(l)}$ for $l = 1, 2, 3$.

Learning proceeds by finding the value of $\boldsymbol{\mu}$ that maximizes the variational lower bound for the current value of model parameters $\theta$, which results in a set of the mean-field fixed-point equations. Given the variational parameters $\boldsymbol{\mu}$, the model parameters $\theta$ are then updated to maximize the variational bound using an MCMC-based stochastic approximation [1, 8, 9].

To initialize the model parameters to good values, we use a greedy layer-wise pretraining strategy by learning a stack of modified RBMs (for details see [1]).

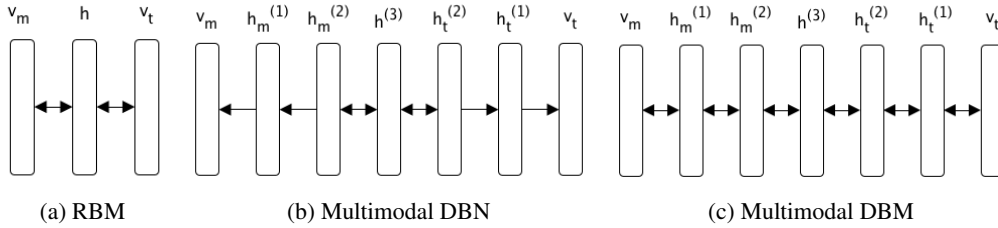

<center>(a) RBM        (b) Multimodal DBN        (c) Multimodal DBM</center>

<center>Figure 3: Different ways of combining multimodal inputs</center>

## 3.2 Salient Features

A Multimodal DBM can be viewed as a composition of unimodal undirected pathways. Each pathway can be pretrained separately in a completely unsupervised fashion, which allows us to leverage a large supply of unlabeled data. Any number of pathways each with any number of layers could potentially be used. The type of the lower-level RBMs in each pathway could be different, accounting for different input distributions, as long as the final hidden representations at the end of each pathway are of the same type.

The intuition behind our model is as follows. Each data modality has very different statistical properties which make it difficult for a single hidden layer model (such as Fig. 3a) to directly find correlations across modalities. In our model, this difference is bridged by putting layers of hidden units between the modalities. The idea is illustrated in Fig. 3c, which is just a different way of displaying Fig. 2. Compared to the simple RBM (Fig. 3a), where the hidden layer $\mathbf{h}$ directly models the distribution over $\mathbf{v}_t$ and $\mathbf{v}_m$, the first layer of hidden units $\mathbf{h}_m^{(1)}$ in a DBM has an easier task to perform - that of modeling the distribution over $\mathbf{v}_m$ and $\mathbf{h}_m^{(2)}$. Each layer of hidden units in the DBM contributes a small part to the overall task of modeling the distribution over $\mathbf{v}_m$ and $\mathbf{v}_t$. In the process, each layer learns successively higher-level representations and removes modality-specific correlations. Therefore, the middle layer in the network can be seen as a (relatively) "modality-free" representation of the input as opposed to the input layers which were "modality-full".

Another way of using a deep model to combine multimodal inputs is to use a Multimodal Deep Belief Network (DBN) (Fig. 3b) which consists of an RBM followed by directed belief networks leading out to each modality. We emphasize that there is an important distinction between this model and the DBM model of Fig. 3c. In a DBN model the responsibility of the multimodal modeling falls entirely on the joint layer. In the DBM, on the other hand, this responsibility is spread out over the entire network. The modality fusion process is distributed across all hidden units in all layers. From the generative perspective, states of low-level hidden units in one pathway can influence the states of hidden units in other pathways through the higher-level layers, which is not the case for DBNs.

## 3.3 Modeling Tasks

**Generating Missing Modalities:** As argued in the introduction, many real-world applications will often have one or more modalities missing. The Multimodal DBM can be used to generate such missing data modalities by clamping the observed modalities at the inputs and sampling the hidden modalities from the conditional distribution by running the standard alternating Gibbs sampler [1].

For example, consider generating text conditioned on a given image[1] $\mathbf{v}_m$. The observed modality $\mathbf{v}_m$ is clamped at the inputs and all hidden units are initialized randomly. $P(\mathbf{v}_t|\mathbf{v}_m)$ is a multinomial distribution over the vocabulary. Alternating Gibbs sampling can be used to sample words from $P(\mathbf{v}_t|\mathbf{v}_m)$. Fig. 1 shows examples of words that have high probability under the conditional distributions.

**Inferring Joint Representations:** The model can also be used to generate a fused representation that multiple data modalities. This fused representation is inferred by clamping the observed modalities and doing alternating Gibbs sampling to sample from $P(\mathbf{h}^{(3)}|\mathbf{v}_m, \mathbf{v}_t)$ (if both modalities are present) or from $P(\mathbf{h}^{(3)}|\mathbf{v}_m)$ (if text is missing). A faster alternative, which we adopt in our experimental results, is to use variational inference (see Sec. 3.1) to approximate posterior $Q(\mathbf{h}^{(3)}|\mathbf{v}_m, \mathbf{v}_t)$ or $Q(\mathbf{h}^{(3)}|\mathbf{v}_m)$. The activation probabilities of hidden units $\mathbf{h}^{(3)}$ constitute the joint representation of the inputs.

This representation can then be used to do information retrieval for multimodal or unimodal queries. Each data point in the database (whether missing some modalities or not) can be mapped to this latent space. Queries can also be mapped to this space and an appropriate distance metric can be used to retrieve results that are close to the query.

**Discriminative Tasks:** Classifiers such as SVMs can be trained with these fused representations as inputs. Alternatively, the model can be used to initialize a feed forward network which can then be finetuned [1]. In our experiments, logistic regression was used to classify the fused representations. Unlike finetuning, this ensures that all learned representations that we compare (DBNs, DBMs and Deep Autoencoders) use the same discriminative model.

## 4 Experiments

### 4.1 Dataset and Feature Extraction

The MIR Flickr Data set [10] was used in our experiments. The data set consists of 1 million images retrieved from the social photography website Flickr along with their user assigned tags. Among the 1 million images, 25,000 have been annotated for 24 topics including object categories such as, *bird*, *tree*, *people* and scene categories, such as *indoor*, *sky* and *night*. A stricter labeling was done for 14 of these classes where an image was annotated with a category only if that category was salient. This leads to a total of 38 classes where each image may belong to several classes. The unlabeled 975,000 images were used only for pretraining. We use 15,000 images for training and 10,000 for testing, following Huiskes et al. [2]. Mean Average Precision (MAP) is used as the performance metric. Results are averaged over 5 random splits of training and test sets.

Each text input was represented using a vocabulary of the 2000 most frequent tags. The average number of tags associated with an image is 5.15 with a standard deviation of 5.13. There are 128,501 images which do not have any tags, out of which 4,551 are in the labeled set. Hence about 18% of the labeled data has images but is missing text. Images were represented by 3857-dimensional features, that were extracted by concatenating Pyramid Histogram of Words (PHOW) features [11], Gist [12] and MPEG-7 descriptors [13] (EHD, HTD, CSD, CLD, SCD). Each dimension was mean-centered and normalized to unit variance. PHOW features are bags of image words obtained by extracting dense SIFT features over multiple scales and clustering them. We used publicly available code ( [14, 15]) for extracting these features.

### 4.2 Model Architecture and Learning

The image pathway consists of a Gaussian RBM with 3857 visible units followed by 2 layers of 1024 hidden units. The text pathway consists of a Replicated Softmax Model with 2000 visible units followed by 2 layers of 1024 hidden units. The joint layer contains 2048 hidden units. Each layer of weights was pretrained using PCD for initializing the DBM model. When learning the DBM model, all word count vectors were scaled so that they sum to 5. This avoids running separate Markov chains for each word count to get the model distribution's sufficient statistics.

Each pathway was pretrained using a stack of modified RBMs. Each Gaussian unit has unit variance that was kept fixed. For discriminative tasks, we perform 1-vs-all classification using logistic regression on the joint hidden layer representation. We further split the 15K training set into 10K for training and 5K for validation.

### 4.3 Classification Tasks

**Multimodal Inputs:** Our first set of experiments, evaluate the DBM as a discriminative model for multimodal data. For each model that we trained, the fused representation of the data was extracted and feed to a separate logistic regression for each of the 38 topics. The text input layer in the DBM was left unclamped when the text was missing. Fig. 4 summarizes the Mean Average Precision (MAP) and precision@50 (precision at top 50 predictions) obtained by different models. Linear Discriminant Analysis (LDA) and Support Vector Machines (SVMs) [2] were trained using the labeled data on concatenated image and text features that did not include SIFT-based features. Hence, to make a fair comparison, our model was first trained using only labeled data with a similar set of features (i.e., excluding our SIFT-based features). We call this model **DBM-Lab**. Fig. 4 shows that the DBM-Lab model already outperforms its competitor SVM and LDA models. DBM-Lab achieves a MAP of 0.526, compared to 0.475 and 0.492, achieved by SVM and LDA models.

| Multimodal Inputs | | |
|---|---|---|
| Model | MAP | Prec@50 |
| Random | 0.124 | 0.124 |
| LDA [2] | 0.492 | 0.754 |
| SVM [2] | 0.475 | 0.758 |
| DBM-Lab | 0.526 | 0.791 |
| DBM-Unlab | 0.585 | 0.836 |
| DBN | 0.599 | 0.867 |
| Autoencoder (based on [5]) | 0.600 | **0.875** |
| DBM | **0.609** | 0.873 |

| Unimodal Inputs | | |
|---|---|---|
| Model | MAP | Prec@50 |
| Image-SVM [2] | 0.375 | - |
| Image-DBN | 0.463 | 0.801 |
| Image-DBM | 0.469 | 0.803 |
| DBM-ZeroText | 0.522 | 0.827 |
| DBM-GenText | **0.531** | **0.832** |

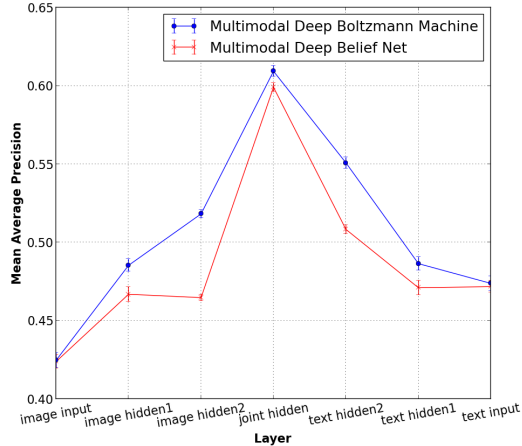

Figure 4: Classification Results. **Left**: Mean Average Precision (MAP) and precision@50 obtained by different models. **Right**: MAP using representations from different layers of multimodal DBMs and DBNs.

To measure the effect of using unlabeled data, a DBM was trained using all the unlabeled examples that had both modalities present. We call this model **DBM-Unlab**. The only difference between the DBM-Unlab and DBM-Lab models is that DBM-Unlab used unlabeled data during its pretraining stage. The input features for both models remained the same. Not surprisingly, the DBM-Unlab model significantly improved upon DBM-Lab achieving a MAP of 0.585. Our third model, **DBM**, was trained using additional SIFT-based features. Adding these features improves the MAP to **0.609**.

We compared our model to two other deep learning models: Multimodal Deep Belief Network (DBN) and a deep Autoencoder model [5]. These models were trained with the same number of layers and hidden units as the DBM. The DBN achieves a MAP of 0.599 and the autoencoder gets 0.600. Their performance was comparable but slightly worse than that of the DBM. In terms of precision@50, the autoencoder performs marginally better than the rest. We also note that the Multiple Kernel Learning approach proposed in Guillaumin et. al. [3] achieves a MAP of 0.623 on the same dataset. However, they used a much larger set of image features (37,152 dimensions).

**Unimodal Inputs:** Next, we evaluate the ability of the model to improve classification of unimodal inputs by filling in other modalities. For multimodal models, the text input was only used during training. At test time, all models were given only image inputs.

Fig. 4 compares the Multimodal DBM model with an SVM over image features alone (**Image-SVM**) [2], a DBN over image features (**Image-DBN**) and a DBM over image features (**Image-DBM**). All deep models had the same depth and same number of hidden units in each layer. Results are reported for two different settings for the Multimodal DBM at test time. In one case (**DBM-ZeroText**), the state of the joint hidden layer was inferred keeping the missing text input clamped at zero. In the other case (**DBM-GenText**), the text input was not clamped and the model was allowed to update the state of the text input layer when performing mean-field updates. In doing so, the model effectively filled-in the missing text modality (some examples of which are shown in Fig. 1). These two settings helped to ascertain the contribution to the improvement that comes from filling in the missing modality.

The DBM-GenText model performs better than all other models, showing that the DBM is able to generate meaningful text that serves as a plausible proxy for missing data. Interestingly, the DBM-ZeroText model does better than any unimodal model. This suggests that *learning multimodal features helps even when some modalities are absent at test time*. Having multiple modalities probably regularizes the model and makes it learn much better features. Moreover, this means that we do not need to learn separate models to handle each possible combination of missing data modalities. One joint model can be deployed at test time and used for any situation that may arise.

Each layer of the DBM provides a different representation of the input. Fig. 4, right panel, shows the MAP obtained by using each of these representations for classification using logistic regression. The input layers, shown at the extreme ends, are not very good at representing useful features. As we go deeper into the model from either input layer towards the middle, the internal representations get better. The joint layer in the middle serves as the most useful feature representation. Observe that the performance of any DBM layer is always better than the corresponding DBN layer, though they get close at the joint layer.

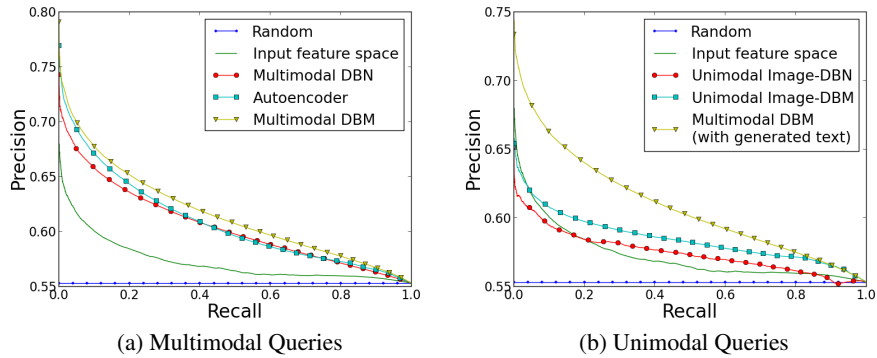

| (a) Multimodal Queries | (b) Unimodal Queries |

Figure 5: Precision-Recall curves for Retrieval Tasks

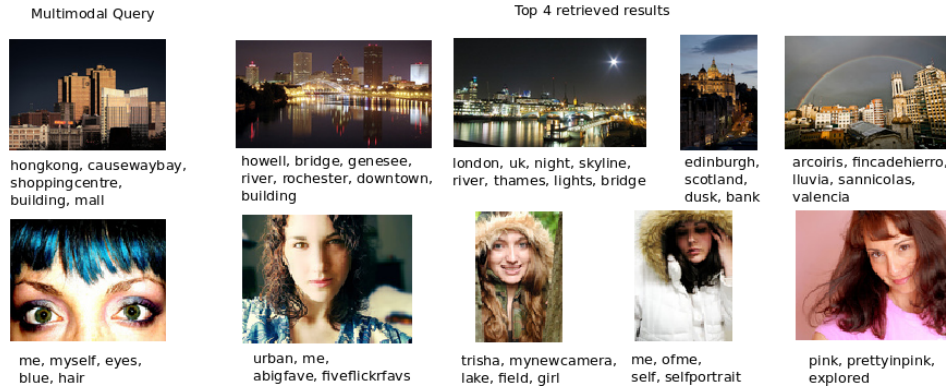

Figure 6: Retrieval Results for Multimodal Queries from the DBM model

## 4.4 Retrieval Tasks

**Multimodal Queries:** The next set of experiments was designed to evaluate the quality of the learned joint representations. A database of images was created by randomly selecting 5000 image-text pairs from the test set. We also randomly selected a disjoint set of 1000 images to be used as queries. Each query contained both image and text modalities. Binary relevance labels were created by assuming that if any of the 38 class labels overlapped between a query and a data point, then that data point is relevant to the query.

Fig. 5a shows the precision-recall curves for the DBM, DBN, and Autoencoder models (averaged over all queries). For each model, all queries and all points in the database were mapped to the joint hidden representation under that model. Cosine similarity function was used to match queries to data points. The DBM model performs the best among the compared models achieving a MAP of 0.622. The autoencoder and DBN models perform worse with a MAP of 0.612 and 0.609 respectively. Fig. 6 shows some examples of multimodal queries and the top 4 retrieved results. Note that even though there is little overlap in terms of text, the model is able to perform well.

**Unimodal Queries:** The DBM model can also be used to query for unimodal inputs by filling in the missing modality. Fig. 5b shows the precision-recall curves for the DBM model along with other unimodal models, where each model received the same image queries as input. By effectively inferring the missing text, the DBM model was able to achieve far better results than any unimodal method (MAP of 0.614 as compared to 0.587 for an Image-DBM and 0.578 for an Image-DBN).

## 5 Conclusion

We proposed a Deep Boltzmann Machine model for learning multimodal data representations. Large amounts of unlabeled data can be effectively utilized by the model. Pathways for each modality can be pretrained independently and "plugged in" together for doing joint training. The model fuses multiple data modalities into a unified representation. This representation captures features that are useful for classification and retrieval. It also works nicely when some modalities are absent and improves upon models trained on only the observed modalities.

**Acknowledgments**: This research was supported by OGS, NSERC and by Early Researcher Award.

## Footnotes

[1]Generating image features conditioned on text can be done in a similar way.

# References

[1] R. R. Salakhutdinov and G. E. Hinton. Deep Boltzmann machines. In *Proceedings of the International Conference on Artificial Intelligence and Statistics*, volume 12, 2009.

[2] Mark J. Huiskes, Bart Thomee, and Michael S. Lew. New trends and ideas in visual concept detection: the MIR flickr retrieval evaluation initiative. In *Multimedia Information Retrieval*, pages 527–536, 2010.

[3] M. Guillaumin, J. Verbeek, and C. Schmid. Multimodal semi-supervised learning for image classification. In *Computer Vision and Pattern Recognition (CVPR), 2010 IEEE Conference on*, pages 902 –909, june 2010.

[4] Eric P. Xing, Rong Yan, and Alexander G. Hauptmann. Mining associated text and images with dual-wing harmoniums. In *UAI*, pages 633–641. AUAI Press, 2005.

[5] Jiquan Ngiam, Aditya Khosla, Mingyu Kim, Juhan Nam, Honglak Lee, and Andrew Y. Ng. Multimodal deep learning. In *International Conference on Machine Learning (ICML)*, Bellevue, USA, June 2011.

[6] Ruslan Salakhutdinov and Geoffrey E. Hinton. Replicated softmax: an undirected topic model. In *NIPS*, pages 1607–1614. Curran Associates, Inc., 2009.

[7] Geoffrey E. Hinton. Training products of experts by minimizing contrastive divergence. *Neural Computation*, 14(8):1711–1800, 2002.

[8] T. Tieleman. Training restricted Boltzmann machines using approximations to the likelihood gradient. In *ICML*. ACM, 2008.

[9] L. Younes. On the convergence of Markovian stochastic algorithms with rapidly decreasing ergodicity rates, March 17 2000.

[10] Mark J. Huiskes and Michael S. Lew. The MIR Flickr retrieval evaluation. In *MIR '08: Proceedings of the 2008 ACM International Conference on Multimedia Information Retrieval*, New York, NY, USA, 2008. ACM.

[11] A Bosch, Andrew Zisserman, and X Munoz. Image classification using random forests and ferns. *IEEE 11th International Conference on Computer Vision (2007)*, 23:1–8, 2007.

[12] Aude Oliva and Antonio Torralba. Modeling the shape of the scene: A holistic representation of the spatial envelope. *International Journal of Computer Vision*, 42:145–175, 2001.

[13] B.S. Manjunath, J.-R. Ohm, V.V. Vasudevan, and A. Yamada. Color and texture descriptors. *Circuits and Systems for Video Technology, IEEE Transactions on*, 11(6):703 –715, 2001.

[14] A. Vedaldi and B. Fulkerson. VLFeat: An open and portable library of computer vision algorithms, 2008.

[15] Muhammet Bastan, Hayati Cam, Ugur Gudukbay, and Ozgur Ulusoy. Bilvideo-7: An mpeg-7-compatible video indexing and retrieval system. *IEEE Multimedia*, 17:62–73, 2010.

